# Generalized Maximum Margin Clustering and Unsupervised Kernel Learning

**Hamed Valizadegan**
Computer Science and Engineering
Michigan State University
East Lansing, MI 48824
valizade@msu.edu

**Rong Jin**
Computer Science and Engineering
Michigan State University
East Lansing, MI 48824
rongjin@cse.msu.edu

## Abstract

Maximum margin clustering was proposed lately and has shown promising performance in recent studies [1, 2]. It extends the theory of support vector machine to unsupervised learning. Despite its good performance, there are three major problems with maximum margin clustering that question its efficiency for real-world applications. First, it is computationally expensive and difficult to scale to large-scale datasets because the number of parameters in maximum margin clustering is quadratic in the number of examples. Second, it requires data preprocessing to ensure that any clustering boundary will pass through the origins, which makes it unsuitable for clustering unbalanced dataset. Third, it is sensitive to the choice of kernel functions, and requires external procedure to determine the appropriate values for the parameters of kernel functions. In this paper, we propose "**generalized maximum margin clustering**" framework that addresses the above three problems simultaneously. The new framework generalizes the maximum margin clustering algorithm by allowing any clustering boundaries including those not passing through the origins. It significantly improves the computational efficiency by reducing the number of parameters. Furthermore, the new framework is able to automatically determine the appropriate kernel matrix without any labeled data. Finally, we show a formal connection between maximum margin clustering and spectral clustering. We demonstrate the efficiency of the generalized maximum margin clustering algorithm using both synthetic datasets and real datasets from the UCI repository.

## 1 Introduction

Data clustering, the unsupervised classification of samples into groups, is an important research area in machine learning for several decades. A large number of algorithms have been developed for data clustering, including the k-means algorithm [3], mixture models [4], and spectral clustering [5, 6, 7, 8, 9]. More recently, maximum margin clustering [1, 2] was proposed for data clustering and has shown promising performance. The key idea of maximum margin clustering is to extend the theory of support vector machine to unsupervised learning. However, despite its success, the following three major problems with maximum margin clustering has prevented it from being applied to real-world applications:

- *High computational cost.* The number of parameters in maximum margin clustering is quadratic in the number of examples. Thus, it is difficult to scale to large-scale datasets. Figure 1 shows the computational time (in seconds) of the maximum margin clustering algorithm with respect to different numbers of examples. We

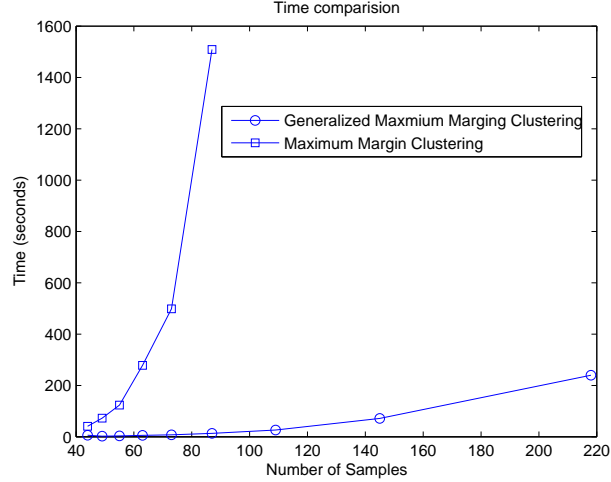

Figure 1: The scalability of the original maximum margin clustering algorithm versus the generalized maximum margin clustering algorithm

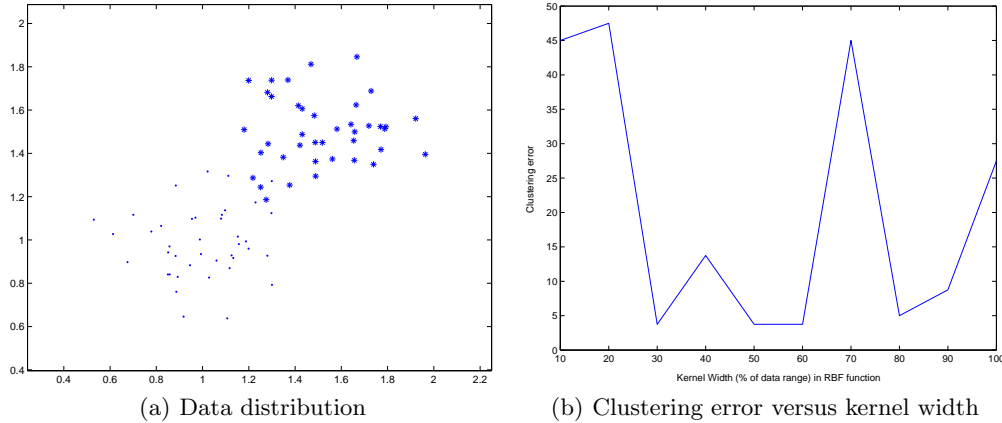

(a) Data distribution        (b) Clustering error versus kernel width

Figure 2: Clustering error of spectral clustering using the RBF kernel with different kernel width. The horizonal axis of Figure 2(b) represents the percentage of the distance range (i.e., the difference between the maximum and the minimum distance) that is used for kernel width.

clearly see that the computational time increases dramatically when we apply the maximum margin clustering algorithm to even modest numbers of examples.

- *Requiring clustering boundaries to pass through the origins.* One important assumption made by the maximum margin clustering in [1] is that the clustering boundaries will pass through the origins. To this end, maximum margin clustering requires centralizing data points around the origins before clustering data. It is important to note that centralizing data points at the origins does not guarantee clustering boundaries to go through origins, particularly when cluster sizes are unbalanced with one cluster significantly more popular than the other.

- *Sensitive to the choice of kernel functions.* Figure 2(b) shows the clustering error of maximum margin clustering for the synthesized data of two overlapped Gaussians clusters (Figure 2(a)) using the RBF kernel with different kernel width. We see that the performance of maximum margin clustering depends critically on the choice of kernel width. The same problem is also observed in spectral clustering [10]. Although a number of studies [8, 9, 10, 6] are devote to automatically identifying appropriate kernel matrices in clustering, they are either heuristic approaches or require additional labeled data.

In this paper, we propose "**generalized maximum margin clustering**" framework that resolves the above three problems simultaneously. In particular, the proposed framework

reformulates the problem of maximum margin clustering to include the bias term in the classification boundary, and therefore remove the assumption that clustering boundaries have to pass through the origins. Furthermore, the new formulism reduces the number of parameters to be linear in the number of examples, and therefore significantly reduces the computational cost. Finally, it is equipped with the capability of unsupervised kernel learning, and therefore, is able to determine the appropriate kernel matrix and clustering memberships simultaneously. More interestingly, we will show that spectral clustering, such as the normalized cut algorithm, can be viewed as a special case of the generalized maximum margin clustering.

The remainder of the paper is organized as follows: Section 2 reviews the work of maximum margin clustering and kernel learning. Section 3 presents the framework of generalized maximum margin clustering. Our empirical studies are presented in Section 4. Section 5 concludes this work.

## 2 Related Work

The key idea of maximum margin clustering is to extend the theory of support vector machine to unsupervised learning. Given the training examples $\mathcal{D} = (\mathbf{x}_1, \mathbf{x}_2, \ldots, \mathbf{x}_n)$ and their class labels $\mathbf{y} = (y_1, y_2, \ldots, y_n) \in \{-1, +1\}^n$, the dual problem of support vector machine can be written as:

$$\max_{\alpha \in \mathbf{R}^n} \quad \alpha^\top \mathbf{e} - \frac{1}{2}\alpha^\top \mathrm{diag}(\mathbf{y})K\mathrm{diag}(\mathbf{y})\alpha$$
$$\text{s. t.} \quad 0 \leq \alpha \leq C, \ \alpha^\top \mathbf{y} = 0 \tag{1}$$

where $K \in \mathbf{R}^{n \times n}$ is the kernel matrix and $\mathrm{diag}(\mathbf{y})$ stands for the diagonal matrix that uses the vector $\mathbf{y}$ as its diagonal elements. To apply the above formulism to unsupervised learning, the maximum margin clustering approach relaxes class labels $\mathbf{y}$ to continuous variables, and searches for both $\mathbf{y}$ and $\alpha$ that maximizes the classification margin. This leads to the following optimization problem:

$$\min_{\mathbf{y}, \lambda, \nu, \delta} \quad t$$
$$\text{s. t.} \quad \begin{pmatrix} (\mathbf{y}\mathbf{y}^\top) \circ K & \mathbf{e} + \nu - \delta + \lambda\mathbf{y} \\ (\mathbf{e} + \nu - \delta + \lambda\mathbf{y})^\top & t - 2C\delta^\top \mathbf{e} \end{pmatrix} \succeq 0$$
$$\nu \geq 0, \ \delta \geq 0$$

where $\circ$ stands for the element wise product between two matrices. To convert the above problem into a convex programming problem, the authors of [1] makes two important relaxations. The first one relaxes $\mathbf{y}\mathbf{y}^\top$ into a positive semi-definitive (PSD) matrix $M \succeq 0$ whose diagonal elements are set to be 1. The second relaxation sets $\lambda = 0$, which is equivalent to assuming that there is no bias term $b$ in the expression of classification boundaries, or in other words, classification boundaries have to pass through the origins of data. These two assumption simplify the above optimization problem as follows:

$$\min_{M, \nu, \delta} \quad t$$
$$\text{s. t.} \quad \begin{pmatrix} M \circ K & \mathbf{e} + \nu - \delta \\ (\mathbf{e} + \nu - \delta)^\top & t - 2C\delta^\top \mathbf{e} \end{pmatrix} \succeq 0$$
$$\nu \geq 0, \ \delta \geq 0, \ M \succeq 0 \tag{2}$$

Finally, a few additional constraints of $M$ are added to the above optimization problem to prevent skewed clustering sizes [1]. As a consequence of these two relaxations, the number of parameters is increased from $n$ to $n^2$, which will significantly increase the computational cost. Furthermore, by setting $\lambda = 0$, the maximum margin clustering algorithm requires clustering boundaries to pass through the origins of data, which is unsuitable for clustering data with unbalanced clusters.

Another important problem with the above maximum margin clustering is the difficulty in determining the appropriate kernel similarity matrix $K$. Although many kernel based clustering algorithms set the kernel parameters manually, there are several studies devoted to automatic selection of kernel functions, in particular the kernel width for the RBF kernel,

i.e., $\sigma$ in $\exp\left(-\frac{\|\mathbf{x}_i - \mathbf{x}_j\|_2^2}{2\sigma^2}\right)$. Shi et al. [8] recommended choosing the kernel width as 10% to 20% of the range of the distance between samples. However, in our experiment, we found that this is not always a good choice, and in many situations it produces poor results. Ng et al. [9] chose kernel width which provides the least distorted clusters by running the same clustering algorithm several times for each kernel width. Although this approach seems to generate good results, it requires running seperate experiments for each kernel width, and therefore could be computationally intensive. Manor et al. in [10] proposed a self-tuning spectral clustering algorithm that computes a different local kernel width for each data point $x_i$. In particular, the local kernel width for each $\mathbf{x}_i$ is computed as the distance of $\mathbf{x}_i$ to its $k$th nearest neighbor. Although empirical study seems to show the effectiveness of this approach, it is unclear how to find the optimal $k$ in computing the local kernel width. As we will see in the experiment section, the clustering accuracy depends heavily on the choice of $k$.

Finally, we will briefly overview the existing work on kernel learning. Most previous work focus on supervised kernel learning. The representative approaches in this category include the kernel alignment [11, 12], semi-definitive programming [13], and spectral graph partitioning [6]. Unlike these approaches, the proposed framework is designed for unsupervised kernel learning.

# 3 Generalized Maximum Margin Clustering and Unsupervised Kernel Learning

We will first present the proposed clustering algorithm for hard margin, followed by the extension to soft margin and unsupervised kernel learning.

## 3.1 Hard Margin

In the case of hard margin, the dual problem of SVM is almost identical to the problem in Eqn. (1) except that the parameter $\alpha$ does not have the upper bound $C$. Following [13], we further convert the problem in (1) into its dual form:

$$\min_{\boldsymbol{\nu}, \mathbf{y}, \lambda} \quad \frac{1}{2}(\mathbf{e} + \nu + \lambda\mathbf{y})^T \text{diag}(\mathbf{y}) K^{-1} \text{diag}(\mathbf{y})(\mathbf{e} + \nu + \lambda\mathbf{y})$$
$$\text{s. t.} \quad \nu \geq 0, \ \mathbf{y} \in \{+1, -1\}^n \tag{3}$$

where $\mathbf{e}$ is a vector with all its elements being one. Unlike the treatment in [13], which rewrites the above problem as a semi-definitive programming problem, we introduce variables $\mathbf{z}$ that is defined as follows:

$$\mathbf{z} = \text{diag}(\mathbf{y})(\mathbf{e} + \nu)$$

Given that $\nu \geq 0$, the above expression for $\mathbf{z}$ is essentially equivalent to the constraint $|z_i| \geq 1$ or $z_i^2 \geq 1$ for $i = 1, 2, \ldots, n$. Then, the optimization problem in (3) is rewritten as follows:

$$\min_{\mathbf{z}, \lambda} \quad \frac{1}{2}(\mathbf{z} + \lambda\mathbf{e})^T K^{-1}(\mathbf{z} + \lambda\mathbf{e})$$
$$\text{s. t.} \quad z_i^2 \geq 1, i = 1, 2, \ldots, n \tag{4}$$

Note that the above problem may not have unique solutions for $\mathbf{z}$ and $\lambda$ due to the translation invariance of the objective function. More specifically, given an optimal solution $\mathbf{z}$ and $\lambda$, we may be able to construct another solution $\mathbf{z}'$ and $\lambda'$ such that:

$$\mathbf{z}' = \mathbf{z} + \epsilon\mathbf{e}, \ \lambda' = \lambda - \epsilon.$$

Evidently, both solutions result in the same value for the objective function in (4). Furthermore, with appropriately chosen $\epsilon$, the new solution $\mathbf{z}'$ and $\lambda'$ will be able to satisfy the constraint $z_i^2 \geq 1$. Thus, $\mathbf{z}'$ and $\lambda'$ is another optimal solution for (3). This is in fact related to the problem in SVM where the bias term $b$ may not be unique [14]. To remove the translation invariance from the objective function, we introduce an additional term $C_e(\mathbf{z}^\top\mathbf{e})^2$ into the objective function, i.e.

$$\min_{\mathbf{z}, \lambda} \quad \frac{1}{2}(\mathbf{z} + \lambda\mathbf{e})^T K^{-1}(\mathbf{z} + \lambda\mathbf{e}) + C_e(\mathbf{z}^\top\mathbf{e})^2$$
$$\text{s. t.} \quad z_i^2 \geq 1, i = 1, 2, \ldots, n \tag{5}$$

where constant $C_e$ weights the important of the punishment factor against the original objective. It is set to be $10,000$ in our experiment. For the simplicity of our expression, we further define

$$\mathbf{w} = (\mathbf{z}; \lambda) \quad \text{and} \quad P = (I_n, \mathbf{e}).$$

Then, the problem in (4) becomes

$$\min_{\mathbf{w} \in \mathbf{R}^{n+1}} \quad \mathbf{w}^T P^T K^{-1} P \mathbf{w} + C_e (\mathbf{e}_0^\top \mathbf{w})^2$$

$$\text{s. t.} \quad w_i^2 \geq 1, i = 1, 2, \ldots, n \tag{6}$$

where $\mathbf{e}_0$ is a vector with all its elements being 1 except its last element which is zero. We then construct the Lagrangian as follows

$$
\begin{aligned}
L(\mathbf{w}, \gamma) &= \mathbf{w}^T P^T K^{-1} P \mathbf{w} + C_e (\mathbf{e}_0^\top \mathbf{w})^2 - \sum_{i=1}^{n} \gamma_i (\mathbf{w}^\top I_{n+1}^i \mathbf{w} - 1) \\
&= \mathbf{w}^\top \left( P^T K^{-1} P + C_e \mathbf{e}_0 \mathbf{e}_0^\top - \sum_{i=1}^{n} \gamma_i I_{n+1}^i \right) \mathbf{w} + \sum_{i=1}^{n} \gamma_i
\end{aligned}
$$

where $I_{n+1}^i$ is an $(n+1) \times (n+1)$ matrix with all the elements being zero except the $i$th diagonal element which is 1. Hence, the dual problem of (6) is

$$\max_{\gamma \in \mathcal{R}^n} \quad \sum_{i=1}^{n} \gamma_i$$

$$\text{s. t.} \quad P^T K^{-1} P + C_e \mathbf{e}_0 \mathbf{e}_0^\top - \sum_{i=1}^{n} \gamma_i I_{n+1}^i \succeq 0$$

$$\gamma_i \geq 0, \ i = 1, 2, \ldots, n \tag{7}$$

Finally, the solution $\mathbf{w}$ can be computed using the KKT condition, i.e.,

$$\left( P^T K^{-1} P + C_e \mathbf{e}_0 \mathbf{e}_0^\top - \sum_{i=1}^{n} \gamma_i I_{n+1}^i \right) \mathbf{w} = \mathbf{0}_{n+1}$$

In other words, the solution $\mathbf{w}$ is proportional to the eigenvector of matrix $\left( P^T K^{-1} P + C_e \mathbf{e}_0 \mathbf{e}_0^\top - \sum_{i=1}^{n} \gamma_i I_{n+1}^i \right)$ for the zero eigenvalue. Since $w_i = (1 + \nu_i) y_i$, $i = 1, 2, \ldots, n$ and $\nu_i \geq 0$, the class labels $\{y_i\}_{i=1}^{n}$ can be inferred directly from the sign of $\{w_i\}_{i=1}^{n}$.

**Remark I**  It is important to realize that the problem in (5) is non-convex due to the non-convex constraint $\mathbf{w}_i^2 \geq 1$. Thus, the optimal solution found by the dual problem in (7) is not necessarily the optimal solution for the prime problem in (5). Our hope is that although the solution found by the dual problem is not optimal for the prime problem, it is still a good solution for the prime problem in (5). This is similar to the SDP relaxation made by the maximum margin clustering algorithm in (2) that relaxes a non-convex programming problem into a convex one. However, unlike the relaxation made in (2) that increases the number of variables from $n$ to $n^2$, the new formulism of maximum margin does not increase the number of parameters (i.e., $\gamma$), and therefore will be computational more efficient. This is shown in Figure 1, in which the computational time of generalized maximum margin clustering is increased much slower than that of the maximum margin algorithm.

**Remark II**  To avoid the high computational cost in estimating $K^{-1}$, we replace $K^{-1}$ with its normalized graph Laplacian $L(K)$ [15], which is defined as $L(K) = I - D^{1/2} K D^{1/2}$ where $D$ is a diagonal matrix whose diagonal elements are computed as $D_{i,i} = \sum_{j=1}^{n} K_{i,j}$, $i = 1, 2, \ldots, n$. This is equivalent to defining a kernel matrix $\tilde{K} = L(K)^\dagger$ where $\dagger$ stands for the operator of pseudo inverse. More interesting, we have the following theorem showing the relationship between generalized maximum margin clustering and the normalized cut.

**Theorem 1.** *The normalized cut algorithm is a special case of the generalized maximum margin clustering in (7) if the following conditions hold, i.e., (1) $K^{-1}$ is set to be the normalized Laplacian $\bar{L}(K)$, (2) all the $\gamma$s are enforced to be the same, i.e., $\gamma_i = \gamma_0, i = 1, 2, \ldots, n$, and (3) $C_e \geq 1$.*

**Proof sketch:** *Given the conditions 1 to 3 in the theorem, the new objective function in (7) becomes:* $\max_{\gamma \geq 0} \gamma$ *s.t.* $\bar{L}(K) \succeq \gamma I_n$ *and the solution for this problem is the largest eigenvector of* $\bar{L}(K)$.

## 3.2 Soft Margin

We extend the formulism in (7) to the case of soft margin by considering the following problem:

$$\min_{\boldsymbol{\nu},\mathbf{y},\lambda,\delta} \quad \frac{1}{2}(\mathbf{e}+\nu-\delta+\lambda\mathbf{y})^T \text{diag}(\mathbf{y})K^{-1}\text{diag}(\mathbf{y})(\mathbf{e}+\nu-\delta+\lambda\mathbf{y}) + C_\delta \sum_{i=1}^{n}\delta_i^2$$

$$\text{s. t.} \quad \nu \geq 0,\ \delta \geq 0,\ \mathbf{y} \in \{+1,-1\}^n \tag{8}$$

where $C_\delta$ weights the importance of the clustering errors against the clustering margin. Similar to the previous derivation, we introduce the slack variable $\mathbf{z}$ and simplify the above problem as follows:

$$\min_{\mathbf{z},\delta,\lambda} \quad \frac{1}{2}(\mathbf{z}+\lambda\mathbf{e})^T K^{-1}(\mathbf{z}+\lambda\mathbf{e}) + C_e(\mathbf{z}^\top \mathbf{e})^2 + C_\delta \sum_{i=1}^{n}\delta_i^2$$

$$\text{s. t.} \quad (z_i+\delta_i)^2 \geq 1, \delta_i \geq 0,\ i=1,2,\ldots,n \tag{9}$$

By approximating $(z_i+\delta_i)^2$ as $z_i^2+\delta_i^2$, we have the dual form of the above problem written as:

$$\max_{\gamma\in\mathcal{R}^n} \quad \sum_{i=1}^{n}\gamma_i$$

$$\text{s. t.} \quad P^\top K^{-1}P + C_e\mathbf{e}_0\mathbf{e}_0^\top - \sum_{i=1}^{n}\gamma_i I_{n+1}^i \succeq 0$$

$$0 \leq \gamma_i \leq C_\delta,\ i=1,2,\ldots,n \tag{10}$$

The main difference between the above formulism and the formulism in (7) is the introduction of the upper bound $C_\delta$ for $\gamma$ in the case of soft margin. In the experiment, we set the parameter $C_\delta$ to be $100,000$, a very large value.

## 3.3 Unsupervised Kernel Learning

As already pointed out, the performance of many clustering algorithms depend on the right choice of the kernel similarity matrix. To address this problem, we extend the formulism in (10) by including the kernel learning mechanism. In particular, we assume that a set of $m$ kernel similarity matrices $K_1, K_2, \ldots, K_m$ are available. Our goal is to identify the linear combination of kernel matrices, i.e., $K = \sum_{i=1}^{m}\beta_i K_i$, that leads to the optimal clustering accuracy. More specifically, we need to solve the following optimization problem:

$$\max_{\gamma,\beta} \quad \sum_{i=1}^{n}\gamma_i$$

$$\text{s. t.} \quad P^\top \left(\sum_{i=1}^{m}\beta_i K_i\right)^{-1} P + C_e\mathbf{e}_0\mathbf{e}_0^\top - \sum_{i=1}^{n}\gamma_i I_{n+1}^i \succeq 0$$

$$0 \leq \gamma_i \leq C_\delta,\ i=1,2,\ldots,n,\ \sum_{i=1}^{m}\beta_i = 1,\ \beta_i \geq 0,\ i=1,2,\ldots,m \tag{11}$$

Unfortunately, it is difficult to solve the above problem due to the complexity introduced by $(\sum_{i=1}^{m}\beta_i K_i)^{-1}$. Hence, we consider an alternative problem to the above one. We first introduce a set of normalized graph Laplacian $\bar{L}_1, \bar{L}_2, \ldots, \bar{L}_m$. Each Laplacian $L_i$ is constructed from the kernel similarity matrix $K_i$. We then defined the inverse of the combined matrix as $K^{-1} = \sum_{i=1}^{m}\beta_i \bar{L}_i$. Then, we have the following optimization problem

$$\max_{\gamma,\beta} \quad \sum_{i=1}^{n}\gamma_i$$

$$\text{s. t.} \quad \sum_{i=1}^{m}\beta_i P^\top \bar{L}_i P + C_e\mathbf{e}_0\mathbf{e}_0^\top - \sum_{i=1}^{n}\gamma_i I_{n+1}^i \succeq 0$$

$$0 \leq \gamma_i \leq C_\delta,\ i=1,2,\ldots,n,\ \sum_{i=1}^{m}\beta_i = 1,\ \beta_i \geq 0,\ i=1,2,\ldots,m \tag{12}$$

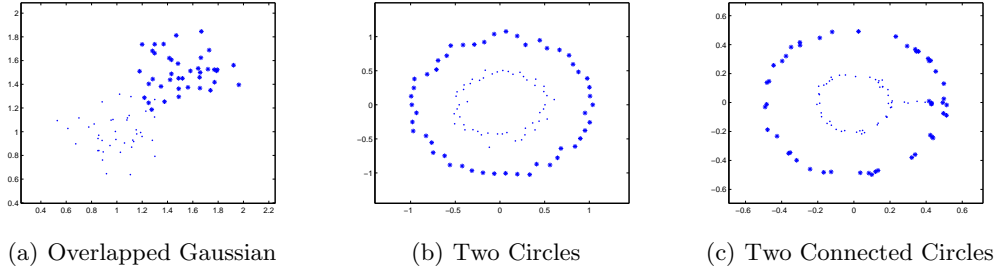

(a) Overlapped Gaussian      (b) Two Circles      (c) Two Connected Circles

Figure 3: Data distribution of the three synthesized datasets

By solving the above problem, we are able to resolve both $\gamma$ (corresponding to clustering memberships) and $\beta$ (corresponding to kernel learning) simultaneously.

## 4   Experiment

We tested the generalized maximum margin clustering algorithm on both synthetic datasets and real datasets from the UCI repository. Figure 3 gives the distribution of the synthetic datasets. The four UCI datasets used in our study are "Vote", "Digits", "Ionosphere", and "Breast". These four datasets comprise of 218, 180, 351, and 285 examples, respectively, and each example in these four datasets is represented by 17, 64, 35, and 32 features. Since the "Digits" dataset consists of multiple classes, we further decompose it into four datasets of binary classes that include pairs of digits difficult to distinguish. Both the normalized cut algorithm [8] and the maximum margin clustering algorithm [1] are used as the baseline. The RBF kernel is used throughout this study to construct the kernel similarity matrices.

In our first experiment, we examine the optimal performance of each clustering algorithm by using the optimal kernel width that is acquired through an exhaustive search. The optimal clustering errors of these three algorithms are summarized in the first three columns of Table 1. It is clear that generalized maximum margin clustering algorithm achieve similar or better performance than both maximum margin clustering and normlized cut for most datasets when they are given the optimal kernel matrices. Note that the results of maximum margin clustering are reported for a subset of samples(including 80 instances) in UCI datasets due to the out of memory problem.

Table 1: Clustering error (%) of normalized cut (NC), maximum margin clustering (MMC), generalized maximum margin clustering (GMMC) and self-tuning spectral clustering (ST).

| Dataset | Optimal Kernel Width | | | Unsupervised Kernel Learning | | |
|---|---|---|---|---|---|---|
| | NC | MMC | GMMC | GMMC | ST (Best $k$) | ST(Worst $k$) |
| Two Circles | 2 | 0 | 0 | 0 | 0 | 50 |
| Two Jointed Circles | 7 | 6.25 | 0 | 0 | 1 | 45 |
| Two Gaussian | 1.25 | 2.5 | 1.25 | 3.75 | 5 | 7.5 |
| Vote | 25 | 15 | 9.6 | 11.90 | 11 | 40 |
| Digits 3-8 | 35 | 10 | 5.6 | 5.6 | 5 | 50 |
| Digits 1-7 | 45 | 31.25 | 2.2 | 3 | 0 | 47 |
| Digits 2-7 | 34 | 1.25 | .5 | 5.6 | 1.5 | 50 |
| Digits 8-9 | 48 | 3.75 | 16 | 12 | 9 | 48 |
| Ionosphere | 25 | 21.25 | 23.5 | 27.3 | 26.5 | 48 |
| Breast | 36.5 | 38.75 | 36.1 | 37 | 37.5 | 41.5 |

In the second experiment, we evaluate the effectiveness of unsupervised kernel learning. Ten kernel matrices are created by using the RBF kernel with the kernel width varied from 10% to 100% of the range of distance between any two examples. We compare the proposed unsupervised kernel learning to the self-tuning spectral clustering algorithm in [10]. One of the problem with the self-tuning spectral clustering algorithm is that its clustering error usually depends on the parameter $k$, i.e., the number of nearest neighbor used for computing the kernel width. To provide a full picture of the self-tuning spectral clustering, we vary $k$ from 1 and 15 , and calculate both best and worst performance using different $k$. The last three columns of Table 1 summarizes the clustering errors of generalized maximum margin

clustering and self-tuning spectral clustering with both best and worst $k$. First, observe the big gap between best and worst performance of self-tuning spectral clustering with different choice of $k$, which implies that this algorithm is sensitive to parameter $k$. Second, for most datasets, generalized maximum margin clustering achieves similar performance as self-tuning spectral clustering with the best $k$. Furthermore, for a number of datasets, the unsupervised kernel learning method achieves the performance close to the one using the optimal kernel width. Both results indicate that the proposed algorithm for unsupervised kernel learning is effective in identifying appropriate kernels.

## 5 Conclusion

In this paper, we proposed a framework for the generalized maximum margin clustering. Compared to the existing algorithm for maximum margin clustering, the new framework has three advantages: 1) it reduces the number of parameters from $n^2$ to $n$, and therefore has a significantly lower computational cost, 2) it allows for clustering boundaries that do not pass through the origin, and 3) it can automatically identify the appropriate kernel similarity matrix through unsupervised kernel learning. Our empirical study with three synthetic datasets and four UCI datasets shows the promising performance of our proposed algorithm.

## References

[1] L. Xu, J. Neufeld, B. Larson, and D. Schuurmans. Maximum margin clustering. In *Advances in Neural Information Processing Systems (NIPS) 17*, 2004.

[2] L. Xu and D. Schuurmans. Unsupervised and semi-supervised multi-class support vector machines. In *Proceedings of the 20th National Conference on Artificial Intelligence (AAAI-05).*, 2005.

[3] J. Hartigan and M. Wong. A k-means clustering algorithm. *Appl. Statist.*, 28:100–108, 1979.

[4] R. A. Redner and H. F. Walker. Mixture densities, maximum likelihood and the em algorithm. *SIAM Review*, 26:195–239, 1984.

[5] C. Ding, X. He, H. Zha, M. Gu, and H. Simon. A min-max cut algorithm for graph partitioning and data clustering. In *Proc. IEEE Int'l Conf. Data Mining*, 2001.

[6] F. R. Bach and M. I. Jordan. Learning spectral clustering. In *Advances in Neural Information Processing Systems 16*, 2004.

[7] R. Jin, C. Ding, and F. Kang. A probabilistic approach for optimizing spectral clustering. In *Advances in Neural Information Processing Systems 18*, 2006.

[8] J. Shi and J. Malik. Normalized cuts and image segmentation. *IEEE Transactions on Pattern Analysis and Machine Intelligence*, 22(8):888–905, 2000.

[9] A. Ng, M. Jordan, and Y. Weiss. On spectral clustering: Analysis and an algorithm. In *Advances in Neural Information Processing Systems 14*, 2001.

[10] L. Zelnik-Manor and P. Perona. Self-tuning spectral clustering. In *Advances in Neural Information Processing Systems 17*, pages 1601–1608, 2005.

[11] N. Cristianini, J. Shawe-Taylor, A. Elisseeff, and J. S. Kandola. On kernel-target alignment. In *NIPS*, pages 367–373, 2001.

[12] X. Zhu, J. Kandola, Z. Ghahramani, and J. Lafferty. Nonparametric transforms of graph kernels for semi-supervised learning. In *Advances in Neural Information Processing Systems 17*, pages 1641–1648, 2005.

[13] G. R. G. Lanckriet, N. Cristianini, P. L. Bartlett, Laurent El Ghaoui, and Michael I. Jordan. Learning the kernel matrix with semidefinite programming. *Journal of Machine Learning Research*, 5:27–72, 2004.

[14] C. J. C. Burges and D. J. Crisp. Uniqueness theorems for kernel methods. *Neurocomputing*, 55(1-2):187–220, 2003.

[15] F.R.K. Chung. *Spectral Graph Theory*. Amer. Math. Society, 1997.
